# A Computer Simulation of Olfactory Cortex With Functional Implications for Storage and Retrieval of Olfactory Information

Matthew A. Wilson and James M. Bower
Computation and Neural Systems Program
Division of Biology, California Institute of Technology, Pasadena, CA 91125

## ABSTRACT

Based on anatomical and physiological data, we have developed a computer simulation of piriform (olfactory) cortex which is capable of reproducing spatial and temporal patterns of actual cortical activity under a variety of conditions. Using a simple Hebb-type learning rule in conjunction with the cortical dynamics which emerge from the anatomical and physiological organization of the model, the simulations are capable of establishing cortical representations for different input patterns. The basis of these representations lies in the interaction of sparsely distributed, highly divergent/convergent interconnections between modeled neurons. We have shown that different representations can be stored with minimal interference, and that following learning these representations are resistant to input degradation, allowing reconstruction of a representation following only a partial presentation of an original training stimulus. Further, we have demonstrated that the degree of overlap of cortical representations for different stimuli can also be modulated. For instance similar input patterns can be induced to generate distinct cortical representations (discrimination), while dissimilar inputs can be induced to generate overlapping representations (accommodation). Both features are presumably important in classifying olfactory stimuli.

## INTRODUCTION

Piriform cortex is a primary olfactory cerebral cortical structure which receives second order input from the olfactory receptors via the olfactory bulb (Fig. 1). It is believed to play a significant role in the classification and storage of olfactory information[1,2,3]. For several years we have been using computer simulations as a tool for studying information processing within this cortex[4,5]. While we are ultimately interested in higher order functional questions, our first modeling objective was to construct a computer simulation which contained sufficient neurobiological detail to reproduce experimentally obtained cortical activity patterns. We believe this first step is crucial both to establish correspondences between the model and the cortex, and to assure that the model is capable of generating output that can be compared to data from actual physiological experiments. In the current case, having demonstrated that the behavior of the simulation at least approximates that of the actual cortex[4] (Fig. 3), we are now using the model to explore the types of processing which could be carried out by this cortical structure. In particular, in this paper we will describe the ability of the simulated cortex to store and recall cortical activity patterns generated by stimulus various conditions. We believe this approach can be used to provide experimentally testable hypotheses concerning the functional organization of this cortex which would have been difficult to deduce solely from neurophysiological or neuroanatomical data.

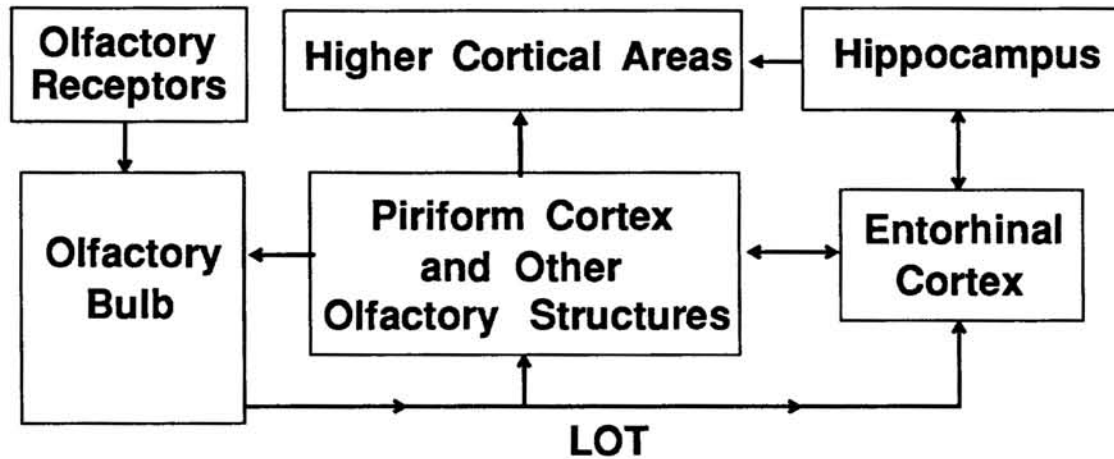

**Fig. 1.** Simplified block diagram of the olfactory system and closely related structures.

## MODEL DESCRIPTION

This model is largely instructed by the neurobiology of piriform cortex[3]. Axonal conduction velocities, time delays, and the general properties of neuronal integration and the major intrinsic neuronal connections approximate those currently described in the actual cortex. However, the simulation reduces both the number and complexity of the simulated neurons (see below). As additional information concerning the these or other important features of the cortex is obtained it will be incorporated in the model. Bracketed numbers in the text refer to the relevent mathematical expressions found in the appendix.

*Neurons.* The model contains three distinct populations of intrinsic cortical neurons, and a fourth set of cells which simulate cortical input from the olfactory bulb (Fig. 2). The intrinsic neurons consist of an excitatory population of pyramidal neurons (which are the principle neuronal type in this cortex), and two populations of inhibitory interneurons. In these simulations each population is modeled as 100 neurons arranged in a 10x10 array (the actual piriform cortex of the rat contains on the order of $10^6$ neurons). The output of each modeled cell type consists of an all-or-none action potential which is generated when the membrane potential of the cell crosses a threshold [2.3]. This output reaches other neurons after a delay which is a function of the velocity of the fiber which connects them and the cortical distance from the originating neuron to each target neuron [2.0, 2.4]. When an action potential arrives at a destination cell it triggers a conductance change in a particular ionic channel type in that cell which has a characteristic time course, amplitude, and waveform [2.0, 2.1]. The effect of this conductance change on the transmembrane potential is to drive it towards the equilibrium potential of that channel. $Na^+$, $Cl^-$, and $K^+$ channels are included in the model. These channels are differentially activated by activity in synapses associated with different cell types (see below).

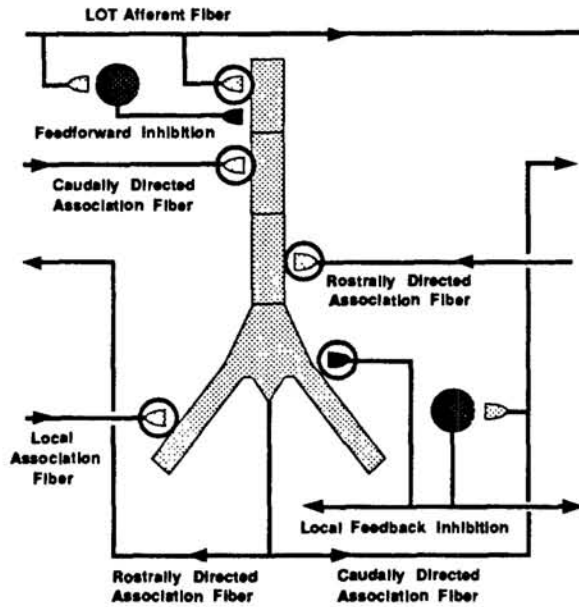

**Fig. 2.** Schematic diagram of piriform cortex showing an excitatory pyramidal cell and two inhibitory interneurons with their local interactions. Circles indicate sites of synaptic modifiability.

*Connection Patterns.* In the olfactory system, olfactory receptors project to the olfactory bulb which, in turn, projects directly to the piriform cortex and other olfactory structures (Fig. 1). The input to the piriform cortex from the olfactory bulb is delivered via a fiber bundle known as the lateral olfactory tract (LOT). This fiber tract appears to make sparse, non-topographic, excitatory connections with pyramidal and feedforward inhibitory neurons across the extent of the cortex[3,6]. In the model this input is simulated as 100 independent cells each of which make random connections (p=0.05) with pyramidal and feedforward inhibitory neurons (Fig. 1 and 2).

In addition to the input connections from the olfactory bulb, there is also an extensive set of connections between the neurons intrinsic to the cortex (Fig. 2). For example, the association fiber system arises from pyramidal cells and makes sparse, distributed excitatory connections with other pyramidal cells all across the cortex[7,8,9]. In the model these connections are randomly distributed with 0.05 probability. In the model and in the actual cortex, pyramidal cells also make excitatory connections with nearby feedforward and feedback inhibitory cells. These interneurons, in turn, make reciprocal inhibitory connections with the group of nearby pyramidal cells. The primary effect of the feedback inhibitory neurons is to inhibit pyramidal cell firing through a Cl⁻ mediated current shunting mechanism[10,11,12]. Feedforward interneurons inhibit pyramidal cells via a long latency, long duration, $K^+$ mediated hyperpolarizing potential[12,13]. Pyramidal cell axons also constitute the primary output of both the model and the actual piriform cortex[7,14].

*Synaptic Properties and Modification Rules.* In the model, each synaptic connection has an associated weight which determines the peak amplitude of the conductance change induced in the postsynaptic cell following presynaptic activity [2.0]. To study learning in the model, synaptic weights associated with some of the fiber systems are modifiable in an activity-dependent fashion (Fig. 2). The basic modification rule in each case is Hebb-like; i.e. change in synaptic strength is proportional to presynaptic activity multiplied by the offset of the postsynaptic membrane potential from a baseline potential. This baseline potential is set slightly more positive than the Cl⁻ equilibrium potential associated with the shunting feedback inhibition. This means that synapses activated while a destination cell is in a depolarized or excited state are strengthened, while those activated during a period of inhibition are weakened. In the model, synapses which follow this rule include the association fiber connections between excitatory pyramidal neurons as well as the connections between inhibitory neurons and pyramidal neurons. Whether these synapses are modifiable in this way in the actual cortex is a subject of active research in our lab. However, the model does mimic the actual synaptic properties associated with the input pathway (LOT) which we have shown to undergo a transient increase in synaptic strength following activation which is independent of postsynaptic potential[15]. This increase is not permanent and the synaptic strength subsequently returns to its baseline value.

*Generation of Physiological Responses.* Neurons in the model are represented as first-order "leaky" integrators with multiple, time-varying inputs [1.0]. During simulation runs, membrane potentials and currents as well as the time of occurence of action potentials are stored for comparison with actual data. An explicit compartmental model (5 compartments) of the pyramidal cells is used to generate the spatial current distributions used for calculation of field potentials (evoked potentials, EEGs) [3.0, 4.0].

*Stimulus Characteristics.* To compare the responses of the model to those of the actual cortex, we mimicked actual experimental stimulation protocols in the simulated cortex and contrasted the resulting intracellular and extracellular records. For example, shock stimuli applied to the LOT are often used to elicit characteristic cortical evoked potentials *in vivo*[16,17,18]. In the model we simulated this stimulus paradigm by simultaneously activating all 100 input fibers. Another measure of cortical activity used most successfully by Freeman and colleagues involves recording EEG activity from piriform cortex in behaving animals[19,20]. These odor-like responses were generated in the model through steady, random stimulation of the input fibers.

To study learning in the model, once physiological measures were established, it was required that we use more refined stimulation procedures. In the absence of any specific information about actual input activity patterns along the LOT, we constructed each stimulus out of a randomly selected set of 10 out of the 100 input fibers. Each stimulus episode consisted of a burst of activity in this subset of fibers with a duration of 10 msec at 25 msec intervals to simulate the 40 Hz periodicity of the actual olfactory bulb input. This pattern of activity was repeated in trials of 200 msec duration which roughly corresponds to the theta rhythm periodicity of bulbar activity and respiration[21,22]. Each trial was then presented 5 times for a total exposure time of 1 second (cortical time). During this period the Hebb-type learning rule could be used to modify the connection weights in an activity-dependent fashion.

*Output Measure for Learning.* Given that the sole output of the cortex is in the form of action potentials generated by the pyramidal cells, the output measure of the model was taken to be the vector of spike frequency for all pyramidal neurons over a 200 msec trial, with each element of the vector corresponding to the firing frequency of a single pyramidal cell. Figures 5 through 8 show the 10 by 10 array of pyramidal cells. The size of the box placed at each cell position represents the magnitude of the spike frequency for that cell. To evaluate learning effects, overlap comparisons between response pairs were made by taking the normalized dot product of their response vectors and expressing that value as a percent overlap (Fig. 4).

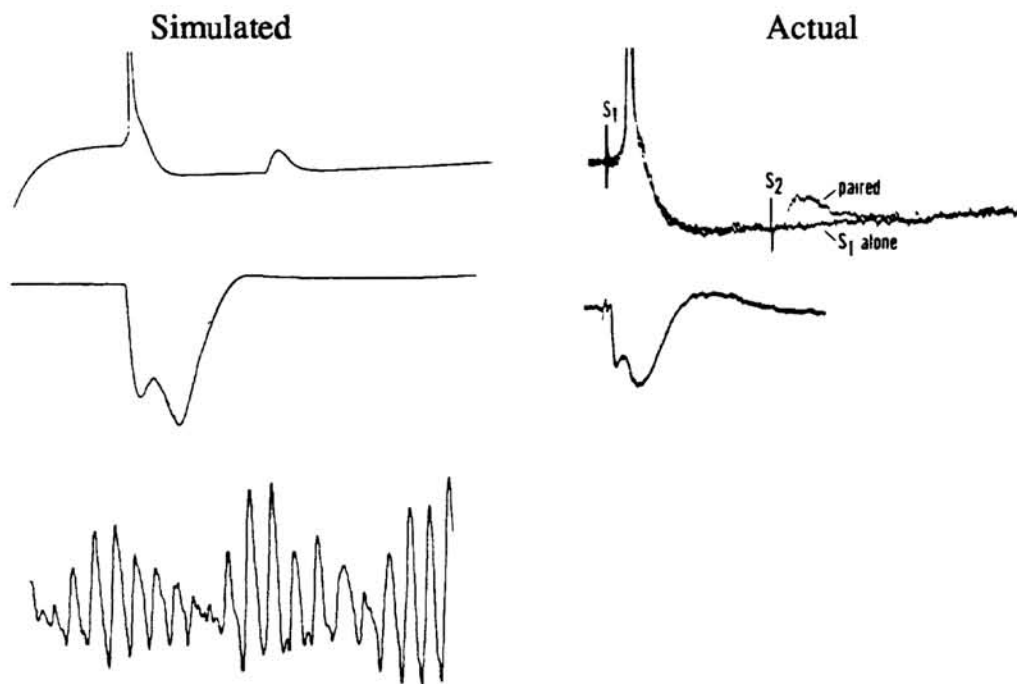

**Fig. 3.** Simulated physiological responses of the model compared with actual cortical responses. Upper: Simulated intracellular response of a single cell to paired stimulation of the input system (LOT) (left) compared with actual response (right) (Haberly & Bower,'84). Middle: Simulated extracellular response recorded at the cortical surface to stimulation of the LOT (left), compared with actual response (right) (Haberly,'73b). Lower: Stimulated EEG response recorted at the cortical surface to odor-like input (left), for actual EEG see Freeman 1978.

*Computational Requirements.* All simulations were carried out on a Sun Microsystems 3/260 model microcomputer equipped with 8 Mbytes of memory and a floating point accelerator. Average time for a 200 msec simulation was 3 cpu minutes.

## RESULTS

### Physiological Responses

As described above, our initial modeling objective was to accurately simulate a wide range of activity patterns recorded, by ourselves and others, in piriform cortex using various physiological procedures. Comparisons between actual and simulated records for several types of response are shown in figure 3. In general, the model replicated known physiological responses quite well (Wilson et al in preparation describes, in detail, the analysis of the physiological results). For example in response to shock stimulation of the input pathway (LOT), the model reproduces the principle characteristics of both the intracellular and location-dependent extracellular waveforms recorded in the actual cortex[9,17,18] (Fig. 3).

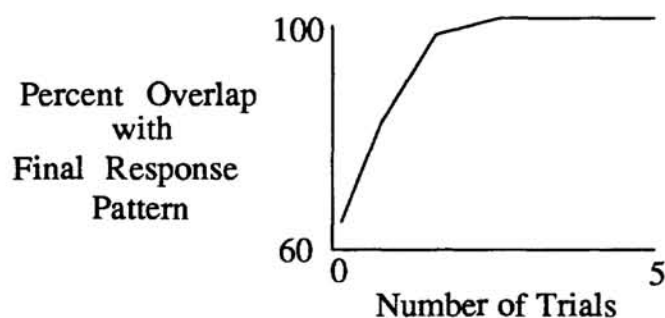

**Fig. 4.** Convergence of the cortical response during training with a single stimulus with synaptic modification.

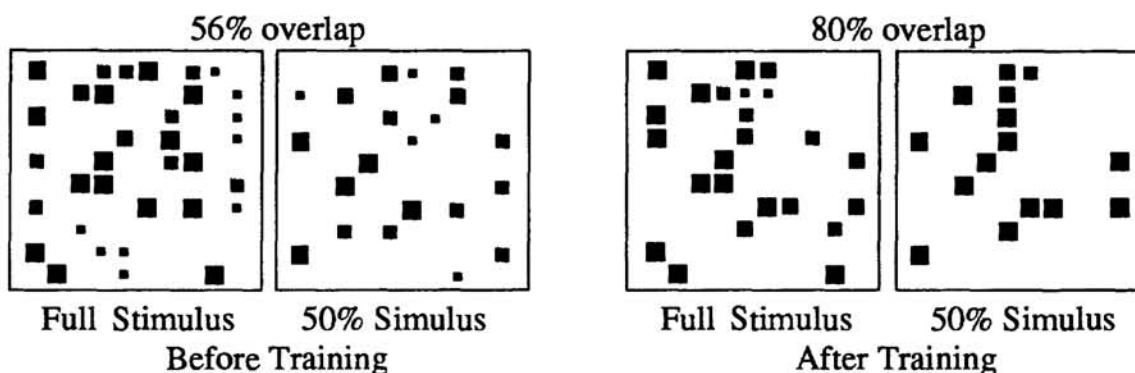

**Fig. 5.** Reconstruction of cortical response patterns with partially degraded stimuli. Left: Response, before training, to the full stimulus (left) and to the same stimulus with 50% of the input fibers inactivated (right). There is a 44% degradation in the response. Right: Response after training, to the full stimulus (left), and to the same stimulus with 50% of the input fibers inactivated (right). As a result of training, the degradation is now only 20%.

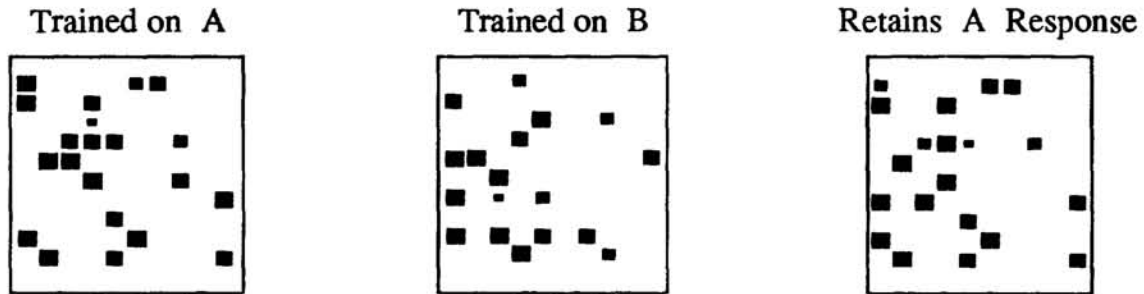

**Fig. 6.** Storage of multiple patterns. Left: Response to stimulus A after training. Middle: Response to stimulus B after training on A followed by training on B. Right: Response to stimulus A after training on A followed by training on B. When compared with the original response (left) there is an 85% congruence.

Further, in response to odor-like stimulation the model exhibits 40 Hz oscillations which are characteristic of the EEG activity in olfactory cortex in awake, behaving animals[19]. Although beyond the scope of the present paper, the simulation also duplicates epileptiform[9] and damped oscillatory[16] type activity seen in the cortex under special stimulus or pharmacological conditions[4].

Learning

Having simulated characteristic physiological responses, we wished to explore the capabilities of the model to store and recall information. Learning in this case is defined as the development of a consistent representation in the activity of the cortex for a particular input pattern with repeated stimulation and synaptic modification. Figure 4 shows how the network converges, with training, on a representation for a stimulus. Having demonstrated that, we studied three properties of learned responses - the reconstruction of trained cortical response patterns with partially degraded stimuli, the simultaneous storage of separate stimulus response patterns, and the modulation of cortical response patterns independent of relative stimulus characteristics.

*Reconstruction of Learned Cortical Response Patterns with Partially Degraded Stimuli.* We were interested in knowing what effect training would have on the sensitivity of cortical responses to fluctuations in the input signal. First we presented the model with a random stimulus A for one trial (without synaptic modification). On the next trial the model was presented with a degraded version of A in which half of the original 10 input fibers were inactivated. Comparison of the responses to these two stimuli in the naive cortex showed a 44% variation. Next, the model was trained on the full stimulus A for 1 second (with synaptic modification). Again, half of the input was removed and the model was presented with the degraded stimulus for 1 trial (without synaptic modification). In this case the dif-

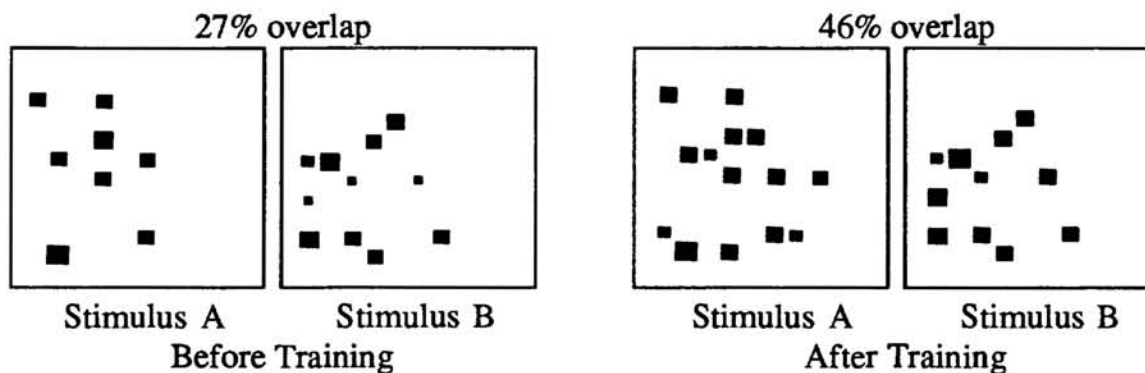

**Fig. 7.** Results of merging cortical response patterns for dissimilar stimuli. Left: Response to stimulus A and stimulus B before training. Stimuli A and B do not activate any input fibers in common but still have a 27% overlap in cortical response patterns. Right: Response to stimulus A and stimulus B after training in the presence of a common modulatory input E1. The overlap in cortical response patterns is now 46%.

ference between cortical responses was only 20% (Fig. 5) showing that training increased the robustness of the response to degradation of the stimulus.

*Storage of Two Patterns.* The model was first trained on a random stimulus A for 1 second. The response vector for this case was saved. Then, continuing with the weights obtained during this training, the model was trained on a new non-overlapping (i.e. different input fibers activated) stimulus B. Both stimulus A and stimulus B alone activated roughly 25% of the cortical pyramidal neurons with 25% overlap between the two responses. Following the second training period we assessed the amount of interference in recalling A introduced by training with B by presenting stimulus A again for a single trial (without synaptic modification). The variation between the response to A following additional training with B and the initially saved reponse to A alone was less than 15% (Fig. 6) demonstrating that learning B did not substantially interfere with the ability to recall A.

*Modulation of Cortical Response Patterns.* It has been previously demonstrated that the stimulus evoked response of olfactory cortex can be modulated by factors not directly tied to stimulus qualities, such as the behavioral state of the animal[1,20,23]. Accordingly we were interested in knowing whether the representations stored in the model could be modulated by the influence of such a "state" input.

One potential role of a "state" input might be to merge the cortical response patterns for dissimilar stimuli; an effect we refer to as accomodation. To test this in the model, we presented it with a random input stimulus A for 1 trial. It was then presented with a random input stimulus B (non-overlapping input fibers). The amount of overlap in the cortical responses for these untrained cases was 27%. Next, the model was trained for 1 second on stimulus A in the presence of an additional random "state" stimulus E1 (activity in a set of 10 input fibers distinct

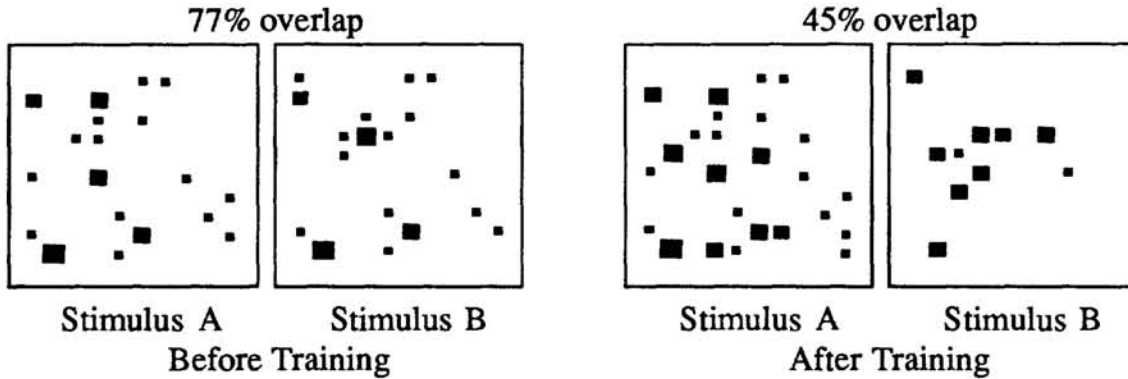

77% overlap                                    45% overlap

Stimulus A        Stimulus B          Stimulus A        Stimulus B
      Before Training                        After Training

**Fig. 8.** Results of differentiating cortical response patterns for similar stimuli. Left: Response to stimulus A and stimulus B before training. Stimuli A and B activate 75% of their input fibers in common and have a 77% overlap in cortical response patterns. Right: Response to stimulus A and stimulus B after training A in the presence of modulatory input E1 and training B with a different modulatory input E2. The overlap in cortical response patterns is now 45%.

from both A and B). The model was then trained on stimulus B in the presence of the same "state" stimulus E1. After training, the model was presented with stimulus A alone for 1 trial and stimulus B alone for 1 trial. Results showed that now, even without the coincident E1 input, the amount of overlap between A and B responses was found to have increased to 46% (Fig 7). The role of E1 in this case was to provide a common stimulus component during learning which reinforced shared components of the responses to input stimuli A and B.

To test the ability of a state stimulus to induce differentiation of cortical response patterns for similar stimuli, we presented the model with a random input stimulus A for 1 trial, followed by 1 trial of a random input stimulus B (75% of the input fibers overlapping). The amount of overlap in the cortical responses for these untrained cases was 77%. Next, the model was trained for a period of 1 second on stimulus A in the presence of an additional random "state" stimulus E1 (a set of 10 input fibers not overlapping either A or B). It was then trained on input stimulus B in the presence of a different random "state" stimulus E2 (10 input fibers not overlapping either A, B, or E1) After this training the model was presented with stimulus A alone for 1 trial and stimulus B alone for 1 trial. The amount of overlap was found to have decreased to 45% (Fig 8). In this situation E1 and E2 provided a differential signal during learning which reinforced distinct components of the responses to input stimuli A and B.

## DISCUSSION

*Physiological Responses.* Detailed discussion of the mechanisms underlying the simulated patterns of physiological activity in the cortex is beyond the scope of the current paper. However, the model has been of value in suggesting roles for

specific features of the cortex in generating physiologically recorded activity. For example, while actual input to the cortex from the olfactory bulb is modulated into 40 Hz bursts[24], continuous stimulation of the model allowed us to demonstrate the model's capability for intrinsic periodic activity independent of the complementary pattern of stimulation from the olfactory bulb. While a similar ability has also been demonstrated by models of Freeman[25], by studying this oscillating property in the model we were able to associate these oscillatory characteristics with specific interactions of local and distant network properties (e.g. inhibitory and excitatory time constants and trans-cortical axonal conduction velocities). This result suggests underlying mechanisms for these oscillatory patterns which may be somewhat different than those previously proposed.

*Learning*. The main subject of this paper is the examination of the learning capabilities of the cortical model. In this model, the apparently sparse, highly distributed pattern of connectivity characteristic of piriform cortex is fundamental to the way in which the model learns. Essentially, the highly distributed pattern of connections allows the model to develop stimulus-specific cortical response patterns by extracting correlations from randomly distributed input and association fiber activity. These correlations are, in effect, stored in the synaptic weights of the association fiber and local inhibitory connections.

The model has also demonstrated robustness of a learned cortical response against degradation of the input signal. A key to this property is the action of sparsely distributed association fibers which provide reinforcment for previously established patterns of cortical activity. This property arises from the modification of synaptic weights due to correlations in activity between intra-cortical association fibers. As a result of this modification the activity of a subset of pyramidal neurons driven by a degraded input drives the remaining neurons in the response.

In general, in the model, similar stimuli will map onto similar cortical responses and dissimilar stimuli will map onto dissimilar cortical responses. However, a presumably important function of the cortex is not simply to store sensory information, but to represent incoming stimuli as a function of the absolute stimulus qualities and the context in which the stimulus occurs. The fact that many of the structures that piriform cortex projects to (and receives projections from) may be involved in multimodal "state" generation[14] is circumstantial evidence that such modulation may occur. We have demonstrated in the model that such a modulatory input can modify the representations generated by pairs of stimuli so as to push the representations of like stimuli apart and pull the representations of dissimilar stimuli together. It should be pointed out that this modulatory input was not an "instructive" signal which explicitly directed the course of the representation, but rather a "state" signal which did not require *a priori* knowledge of the representational structure. In the model, this modulatory phenomenon is a simple consequence of the degree of overlap in the combined (odor stimulus + modulator) stimulus. Both cases approached approximately 50% overlap in cortical responses reflecting the approximately 50% overlap in the combined stimuli for both cases.

Of interest was the use of the model's reconstructive capabilities to maintain the modulated response to each input stimulus even in the absence of the modulatory input.

## CAVEATS AND CONCLUSIONS

Our approach to studying this system involves using computer simulation to investigate mechanisms of information processing which could be implemented given what is known about biological constraints. The significance of results presented here lies primarily in the finding that the structure of the model and the parameter settings which were appropriate for the reproduction of physiological responses were also appropriate for the proper convergence of a simple, biologically plausible learning rule under various conditions. Of course, the model we have developed is only an approximation to the actual cortex limited by our knowledge of its organization and the computing power available. For example, the actual piriform cortex of the rat contains on the order of $10^6$ cells (compared with $10^2$ in the simulations) with a sparsity of connection on the order of p=0.001 (compared with p=0.05 in the simulations). Our continuing research effort will include explorations of the scaling properties of the network.

Other assumptions made in the context of the current model include the assumption that the representation of information in piriform cortex is in the form of spatial distributions of rate-coded outputs. Information contained in the spatio-temporal patterns of activity was not analyzed, although preliminary observation suggests that this may be of significance. In fact, the dynamics of the model itself suggest that temporally encoded information in the input at various time scales may be resolvable by the cortex. Additionally, the output of the cortex was assumed to have spatial uniformity, i.e. no differential weighting of information was made on the basis of spatial location in the cortex. But again, observation of the dynamics of the model, as well as the details of known anatomical distribution patterns for axonal connections, indicate that this is a major oversimplification. Preliminary evidence from the model would indicate that some form of hierarchical structuring of information along rostral/caudal lines may occur. For example it may be that cells found in progressively more rostral locations would have increasingly non-specific odor responses.

Further investigations of learning within the model will explore each of these issues more fully, with attempts to correlate simulated findings with actual recordings from awake, behaving animals. At the same time, new data pertaining to the structure of the cortex will be incorporated into the model as it emerges.

## ACKNOWLEDGEMENTS

We wish to thank Dr. Lewis Haberly and Dr. Joshua Chover for their roles in the development and continued support of the modeling effort. We also wish to thank Dave Bilitch for his technical assistance. This work was supported by NIH grant NS22205, NSF grant EET-8700064, the Lockheed Corporation, and a fellowship from the ARCS foundation.

# APPENDIX

$$\frac{dV_i}{dt} = \frac{1}{c_m}\left[\sum_{k=1}^{n_{types}} I_{ik}(t) + \frac{E_r - V_i(t)}{r_l}\right] \qquad (1.0)$$

*Somatic Integration*

$$I_{ik}(t) = [E_k - V_i(t)]g_{ik}(t) \qquad (1.1)$$

$n_{types}$ = number of input types
$V_i(t)$ = membrane potential of $i$th cell
$I_{ik}(t)$ = current into cell $i$ due to input type $k$
$E_k$ = equilibrium potential associated with input type $k$

$E_r$ = resting potential
$r_l$ = membrane leakage resistance
$c_m$ = membrane capacitance
$g_{ik}(t)$ = conductance due to input type $k$ in cell $i$

$$g_{ik}(t) = \sum_{j=1}^{n_{cells}} \int_{\lambda=0}^{\lambda=d_k} F_k(\lambda)\, A_{ijk}\, W_{ij}\, S_j\left(t - \lambda - \frac{L_{ij}}{v_k} - \varepsilon_k\right) d\lambda \qquad (2.0)$$

$$F_k(t) = \frac{t}{\tau} e^{(1-\frac{t}{\tau})}\left[(1-U(t-\tau)) + U(t-\tau)\cos\left[\frac{\pi}{2}\frac{(t-\tau)}{(d_k-\tau)}\right]\right], \quad \tau = \gamma d_k \qquad (2.1)$$

*Spike Propagation*

*and Synaptic Input*

$$A_{ijk} = (1-\rho_k^{min})e^{-L_{ij}\rho_k} + \rho_k^{min} \qquad (2.2)$$

$$S_j(t) = \begin{cases} 1 & V_j(t)>T_j, \quad S_j(\lambda)=0 \text{ for } \lambda=t..t-\Delta t_r \\ 0 & \text{otherwise} \end{cases} \qquad (2.3)$$

$$L_{ij} = |i - j|\Delta x \qquad (2.4)$$

$n_{cells}$ = number of cells in the simulation
$\Delta x$ = distance between adjacent cells
$d_k$ = duration of conductance change due to input type $k$
$v_k$ = velocity of signals for input type $k$
$\varepsilon_k$ = latency for input type $k$
$\rho_k$ = spatial attenuation factor for input type $k$
$\rho_k^{min}$ = minimum spatial attenuation for input type $k$
$\Delta t_r$ = refractory period

$T_j$ = threshold for cell $j$
$L_{ij}$ = distance from cell $i$ to cell $j$
$A_{ijk}$ = distribution of synaptic density for input type $k$
$W_{ij}$ = synaptic weight from cell $j$ to cell $i$
$g_{ik}(t)$ = conductance due to input type $k$ in cell $i$
$F_k(t)$ = conductance waveform for input type $k$
$S_j(t)$ = spike output of cell $j$ at time $t$
$U(t)$ = unit step function

*Field Potentials*

$$V_{ep}^j(t) = \frac{R_e}{4\pi}\sum_{i=1}^{n_{cells}}\sum_{n=1}^{n_{segs}} \frac{I_m^{in}(t)}{\left[(z_{elec}-z_n)^2 + (x_j-x_i)^2\right]^{\frac{1}{2}}} \qquad (3.0)$$

$n_{cells}$ = number of cells in the simulation
$n_{segs}$ = number of segments in the compartmental model
$V_{ep}^j(t)$ = approximate extracellular field potential at cell $j$
$I_m^{in}(t)$ = membrane current for segment $n$ in cell $i$

$z_{elec}$ = depth of recording site
$z_n$ = depth of segment $n$
$x_j$ = x location of the $j$th cell
$R_e$ = extracellular resistance per unit length

$$\frac{dV_n}{dt} = \frac{1}{c_m^n}\left[I_a^{n-}(t) + I_a^{n+}(t) + \frac{E_r - V_n(t)}{r_m^n} + \sum_{c=1}^{n_{chan}} [E_c - V_n(t)]g_{nc}(t)\right] \qquad (4.0)$$

*Dendritic Model*

$$I_a^{n-}(t) = \frac{V_{n-1}(t) - V_n(t)}{r_a^{n-1} + r_a^n}, \qquad I_a^{n+}(t) = \frac{V_{n+1}(t) - V_n(t)}{r_a^{n+1} + r_a^n} \qquad (4.1)$$

$$I_m^n(t) = I_a^{n-}(t) + I_a^{n+}(t) \tag{4.2}$$

$$r_a^n = \frac{1}{2}\left[R_e l_n + R_i \frac{l_n}{\pi\left(\frac{d_n}{2}\right)^2}\right] \quad , \quad r_m^n = \frac{R_m}{\pi\, l_n d_n} \quad , \quad c_m^n = C_m \pi\, l_n d_n \tag{4.3}$$

$n_{chan}$ = number of different channels per segment
$V_n(t)$ = membrane potential of $n$ th segment
$c_m^n$ = membrane capacitance for segment $n$
$r_a^n$ = axial resistance for segment $n$
$r_m^n$ = membrane resistance for segment $n$
$g_{nc}(t)$ = conductance of channel $c$ in segment $n$
$E_c$ = equilibrium potential associated with channel $c$
$I_a^{n\pm}(t)$ = axial current between segment $n\pm1$ and $n$

$I_m^n(t)$ = membrane current for segment $n$
$l_n$ = length of segment $n$
$d_n$ = diameter of segment $n$
$R_m$ = membrane resistivity
$R_i$ = intracellular resistivity per unit length
$R_e$ = extracellular resistance per unit length
$C_m$ = capacitance per unit surface area

# REFERENCES

1. W. J. Freeman, J. Neurophysiol., 23, 111 (1960).
2. T. Tanabe, M. Iino, and S. F. Takagi, J. Neurophysiol., 38,1284 (1975).
3. L. B. Haberly, Chemical Senses, 10, 219 (1985).
4. M. Wilson, J. M. Bower, J. Chover, and L. B. Haberly, Soc. Neuro. Abs., 11, 317 (1986).
5. M. Wilson and J. M. Bower, Soc. Neurosci. Abs., 12, 310 (1987).
6. M. Devor, J. Comp. Neur., 166, 31 (1976).
7. L. B. Haberly and J. L. Price, J. Comp. Neurol., 178, 711 (1978a).
8. L. B. Haberly and S. Presto, J. Comp. Neur., 248, 464 (1986).
9. L. B. Haberly and J. M. Bower, J. Neurophysiol., 51, 90 (1984).
10. M. A. Biedenbach and C. F. Stevens, J. Neurophysiol., 32, 193 (1969).
11. M. A. Biedenbach and C. F. Stevens, J. Neurophysiol., 32, 204 (1969).
12. M. Satou, K. Mori, Y. Tazawa, and S. F. Takagi, J. Neurophysiol., 48, 1157 (1982).
13. G. F. Tseng and L. B. Haberly, Soc. Neurosci. Abs. 12, 667 (1986).
14. L. B. Luskin and J. L. Price, J. Comp. Neur., 216, 264 (1983).
15. J. M. Bower and L. B. Haberly,L.B., Proc. Natl. Acad. Sci. USA, 83, 1115 (1985).
16. W. J. Freeman, J. Neurophysiol., 31, 1 (1968).
17. L. B. Haberly, J. Neurophysiol., 36, 762 (1973).
18. L. B. Haberly, J. Neurophysiol., 36, 775 (1973).
19. W. J. Freeman, Electroenceph. and Clin. Neurophysiol., 44, 586 (1978).
20. W.J. Freeman and W. Schneider, Psychophysiology, 19, 44 (1982).
21. F. Macrides and S. L. Chorover, Science, 175, 84 (1972).
22. F. Macrides, H. B. Eigenbaum, and W. B. Forbes, J. Neurosci., 2, 12, 1705 (1982).
23. P. D. MacLean, N. H. Horwitz, and F. Robinson, Yale J. Biol. Med., 25, 159 (1952).
24. E. D. Adrian, Electroenceph. and Clin. Neurophysiol., 2, 377 (1950).
25. W. J. Freeman, Exp. Neurol., 10, 525 (1964).
